# Online allocation and homogeneous partitioning for piecewise constant mean-approximation

**Odalric Ambrym Maillard**
Montanuniversität Leoben
Franz-Josef Strasse 18
A-8700 Leoben, Austria
odalricambrym.maillard@gmail.com

**Alexandra Carpentier**
Statistical Laboratory, CMS
Wilberforce Road, Cambridge
CB3 0WB UK
a.carpentier@statslab.cam.ac.uk

## Abstract

In the setting of active learning for the multi-armed bandit, where the goal of a learner is to estimate with equal precision the mean of a finite number of arms, recent results show that it is possible to derive strategies based on finite-time confidence bounds that are competitive with the best possible strategy. We here consider an extension of this problem to the case when the arms are the cells of a finite partition $\mathcal{P}$ of a continuous sampling space $\mathcal{X} \subset \mathbb{R}^d$. Our goal is now to build a piecewise constant approximation of a noisy function (where each piece is one region of $\mathcal{P}$ and $\mathcal{P}$ is fixed beforehand) in order to maintain the local quadratic error of approximation on each cell equally low. Although this extension is not trivial, we show that a simple algorithm based on upper confidence bounds can be proved to be adaptive to the function itself in a near-optimal way, when $|\mathcal{P}|$ is chosen to be of minimax-optimal order on the class of $\alpha-$Hölder functions.

## 1 Setting and Previous work

Let us consider some space $\mathcal{X} \subset \mathbb{R}^d$, and $\mathcal{Y} \subset \mathbb{R}$. We call $\mathcal{X}$ the input space or sampling space, $\mathcal{Y}$ the output space or value space. We consider the problem of estimating with uniform precision the function $f : \mathcal{X} \subset \mathbb{R}^d \to \mathcal{Y} \subset \mathbb{R}$. We assume that we can query $n$ times the function $f$, anywhere in the domain, and observe noisy samples of this function. These samples are collected sequentially, and our aim is to design an adaptive procedure that selects wisely *where* on the domain to query the function, according to the information provided by the previous samples. More formally:

**Observed process** We consider an unknown $\mathcal{Y}$-valued process defined on $\mathcal{X}$, written $\nu : \mathcal{X} \to \mathfrak{M}_1^+(\mathcal{Y})$, where $\mathfrak{M}_1^+(\mathcal{Y})$ refers to the set of all probability measures on $\mathcal{Y}$, such that for all $x \in \mathcal{X}$, the random variable $Y(x) \sim \nu(x)$ has mean $f(x) \stackrel{\text{def}}{=} \mathbb{E}[Y(x)|x] \in \mathbb{R}$. We write for convenience the model in the following way

$$Y(x) = f(x) + \mathbf{noise}(x) \,,$$

where $\mathbf{noise}(x) \stackrel{\text{def}}{=} Y(x) - \mathbb{E}[Y(x)|x]$ is the centered random variable corresponding to the noise, with unknown variance $\sigma^2(x)$. We assume throughout this paper that $f$ is $\alpha$-Hölder.

**Partition** We consider we can define a partition $\mathcal{P}$ of the input space $\mathcal{X}$, with finitely many $P$ regions $\{\mathcal{R}_p\}_{1 \leq p \leq P}$ that are assumed to be convex and not degenerated, i.e. such that the interior of each region $\mathcal{R}_p$ has positive Lebesgue volume $\mathbf{v}_p$. Moreover, with each region $\mathcal{R}_p$ is associated a sampling distribution in that region, written $\mu_p \in \mathfrak{M}_1^+(\mathcal{R}_p)$. Thus, when we decide to sample in region $\mathcal{R}_p$, a new sample $X \in \mathcal{R}_p$ is generated according to $X \sim \mu_p$.

**Allocation.** We consider that we have a finite budget of $n \in \mathbb{N}$ samples that we can use in order to allocate samples as we wish among the regions $\{\mathcal{R}_p\}_{1 \leq p \leq P}$. For illustration, let us assume that we deterministically allocate $T_{p,n} \in \mathbb{N}$ samples in region $\mathcal{R}_p$, with the constraint that the allocation $\{T_{p,n}\}_{1 \leq p \leq P}$ must some to $n$. In region $\mathcal{R}_p$, we thus sample points $\{X_{p,i}\}_{1 \leq p \leq P}$ at random

according to the sampling distribution $\mu_p$, and then get the corresponding values $\{Y_{p,i}\}_{1 \leq i \leq T_{p,n}}$, where $Y_{p,i} \sim \nu(X_{p,i})$. In the sequel, the distribution $\mu_p$ is assumed to be the uniform distribution over region $\mathcal{R}_p$, i.e. the density of $\mu_p$ is $\frac{d\lambda(x) \mathbf{1}_{x \in \mathcal{R}_p}}{\lambda(\mathcal{R}_p)}$ where $\lambda$ denotes the Lebesgue measure. Note that this is not restrictive since we are in an active, not passive setting.

**Piecewise constant mean-approximation.** We use the collected samples in order to build a piecewise constant approximation $\hat{f}_n$ of the mean $f$, and measure the accuracy of approximation on a region $\mathcal{R}_p$ with the expected quadratic norm of the approximation error, namely

$$\mathbb{E}\left[ \int_{\mathcal{R}_p} (f(x) - \hat{f}_n(x))^2 \frac{\lambda(dx)}{\lambda(\mathcal{R}_p)} \right] = \mathbb{E}_{\mu_p, \nu}\left[ (f(X) - \hat{m}_{p,n})^2 \right],$$

where $\hat{m}_{p,n}$ is the constant value that takes $\hat{f}_n$ on the region $\mathcal{R}_p$. A natural choice for the estimator $\hat{m}_{p,n}$ is to use the empirical mean that is unbiased and asymptotically optimal for this criterion. Thus we consider the following estimate (histogram)

$$\hat{f}_n(x) = \sum_{p=1}^{P} \hat{m}_{p,n} \mathbb{I}\{x \in \mathcal{R}_p\} \quad \text{where} \quad \hat{m}_{p,n} = \frac{1}{T_{p,n}} \sum_{i=1}^{T_{p,n}} Y_{p,i} \,.$$

**Pseudo loss** Note that, since the $T_{p,n}$ are deterministic, the expected quadratic norm of the approximation error of this estimator can be written in the following form

$$
\begin{aligned}
\mathbb{E}_{\mu_p, \nu}\left[ (f(X) - \hat{m}_{p,n})^2 \right] &= \mathbb{E}_{\mu_p, \nu}\left[ (f(X) - \mathbb{E}_{\mu_p}[f(X)])^2 \right] + \mathbb{E}_{\mu_p, \nu}\left[ (\mathbb{E}_{\mu_p}[f(X)] - \hat{m}_{p,n})^2 \right] \\
&= \mathbb{V}_{\mu_p}\left[ f(X) \right] + \mathbb{V}_{\mu_p, \nu}\left[ \hat{m}_{p,n} \right] \\
&= \mathbb{V}_{\mu_p}\left[ f(X) \right] + \frac{1}{T_{p,n}} \mathbb{V}_{\mu_p, \nu}\left[ Y(X) \right].
\end{aligned}
$$

Now, using the following immediate decomposition

$$\mathbb{V}_{\mu_p, \nu}\left[ Y(X) \right] = \mathbb{V}_{\mu_p}\left[ f(X) \right] + \int_{\mathcal{R}_p} \sigma^2(x) \mu_p(dx) \,,$$

we deduce that the maximal expected quadratic norm of the approximation error over the regions $\{\mathcal{R}_p\}_{1 \leq p \leq P}$, that depends on the choice of the considered allocation strategy $\mathcal{A} \stackrel{\text{def}}{=} \{T_{p,n}\}_{1 \leq p \leq P}$ is thus given by the following so-called pseudo-loss

$$\mathcal{L}_n(\mathcal{A}) \stackrel{\text{def}}{=} \max_{1 \leq p \leq P} \left\{ \frac{T_{p,n} + 1}{T_{p,n}} \mathbb{V}_{\mu_p}\left[ f(X) \right] + \frac{1}{T_{p,n}} \mathbb{E}_{\mu_p}\left[ \sigma^2(X) \right] \right\}. \tag{1}$$

Our goal is to minimize this pseudo-loss. Note that this is a local measure of performance, as opposed to a more usual yet less challenging global quadratic error. Eventually, as the number of cells tends to $\infty$, this local measure of performance approaches $\sup_{x \in \mathcal{X}} \mathbb{E}_\nu\left[ (f(x) - \hat{f}_n(x))^2 \right]$. At this point, let us also introduce, for convenience, the notation $Q_p(T_{p,n})$ that denotes the term inside the max, in order to emphasize the dependency on the quadratic error with the allocation.

**Previous work**

There is a huge literature on the topic of functional estimation in *batch setting*. Since it is a rather old and well studied question in statistics, many books have been written on this topic, such as Bosq and Lecoutre [1987], Rosenblatt [1991], Györfi et al. [2002], where piecewise constant mean-approximation are also called "partitioning estimate" or "regressogram" (first introduced by Tukey [1947]). The minimax-optimal rate of approximation on the class of $\alpha$-Hölder functions is known to be in $O(n^{-\frac{2\alpha}{2\alpha+d}})$ (see e.g. Ibragimov and Hasminski [1981], Stone [1980], Györfi et al. [2002]). In such setting, a dataset $\{(X_i, Y_i)\}_{i \leq n}$ is given to the learner, and a typical question is thus to try to find the best possible histogram in order to minimize a approximation error. Thus the dataset is fixed and we typically resort to techniques such as model selection where each model corresponds to one histogram (see Arlot [2007] for an extensive study of such).

However, we here ask a very different question, that is how to optimally sample in an *online setting* in order to minimize the approximation error of some histogram. Thus we choose the histogram

before we see any sample, then it is fixed and we need to decide which cell to sample from at each time step. Motivation for this setting comes naturally from some recent works in the setting of active learning for the multi-armed bandit problem Antos et al. [2010], Carpentier et al. [2011]. In these works, the objective is to estimate with equal precision the mean of a finite number of distributions (arms), which would correspond to the special case when $\mathcal{X} = \{1, \ldots, P\}$ is a finite set in our setting. Intuitively, we reduce the problem to such bandit problem with finite set of arms (regions), and our setting answers the question whether it is possible to extend those results to the case when the arms do not correspond to a singleton, but rather to a continuous region. We show that the answer is positive, yet non trivial. This is non trivial due to the variance estimation in each region: points x in some region may have different means f(x), so that standard estimators for the variance are biased, contrary to the point-wise case and thus finite-arm techniques may yield disastrous results. (Estimating the variance of the distribution in a continuous region actually needs to take into account not only the point-wise noise but also the variation of the function $f$ and the noise level $\sigma^2$ in that region.) We describe a way, inspired from quasi Monte-Carlo techniques, to correct this bias so that we can handle the additional error. Also, it is worth mentioning that this setting can be informally linked to a notion of curiosity-driven learning (see Schmidhuber [2010], Baranes and Oudeyer [2009]), since we want to decide in which region of the space to sample, without explicit reward but optimizing the goal to understand the unknown environment.

**Outline** Section 2 provides more intuition about the pseudo-loss and a result about the optimal oracle strategy when the domain is partitioned in a minimax-optimal way on the class of $\alpha-$Hölder functions. Section 3 presents our assumptions, that are basically to have a sub-Gaussian noise and smooth mean and variance functions, then our estimator of the pseudo-loss together with its concentration properties, before introducing our sampling procedure, called OAHPA-pcma. Finally, the performance of this procedure is provided and discussed in Section 4.

## 2 The pseudo-loss: study and optimal strategies

### 2.1 More intuition on each term in the pseudo-loss

It is natural to look at what happens to each of the two terms that appear in equation 1 when one makes $\mathcal{R}_p$ shrink towards a point. More precisely, let $x_p$ be the mean of $X \sim \mu_p$ and let us look at the limit of $\mathbb{V}_{\mu_p}(f(X))$ when $\mathbf{v}_p$ goes to 0. Assuming that $f$ is differentiable, we get

$$
\begin{aligned}
\lim_{\mathbf{v}_p \to 0} \mathbb{V}_{\mu_p}(f(X)) &= \lim_{\mathbf{v}_p \to 0} \mathbb{E}_{\mu_p}\left[\left(f(X) - f(x_p) - \mathbb{E}[f(X) - f(x_p)]\right)^2\right] \\
&= \lim_{\mathbf{v}_p \to 0} \mathbb{E}_{\mu_p}\left[\left(\langle X - x_p, \nabla f(x_p)\rangle - \mathbb{E}[\langle X - x_p, \nabla f(x_p)\rangle]\right)^2\right] \\
&= \lim_{\mathbf{v}_p \to 0} \mathbb{E}_{\mu_p}\left[\langle X - x_p, \nabla f(x_p)\rangle^2\right] \\
&= \lim_{\mathbf{v}_p \to 0} \nabla f(x_p)^T \mathbb{E}_{\mu_p}\left[(X - x_p)(X - x_p)^T\right]\nabla f(x_p).
\end{aligned}
$$

Therefore, if we introduce $\Sigma_p$ to be the covariance matrix of the random variable $X \sim \mu_p$, then we simply have $\lim_{\mathbf{v}_p \to 0} \mathbb{V}_{\mu_p}(f(X)) = \lim_{\mathbf{v}_p \to 0} ||\nabla f(x_p)||^2_{\Sigma_p}$.

**Example with hyper-cubic regions** An important example is when $\mathcal{R}_p$ is a hypercube with side length $\mathbf{v}_p^{1/d}$ and $\mu_p$ is the uniform distribution over the region $\mathcal{R}_p$. In that case (see Lemma 1), we have $\mu_p(dx) = \dfrac{dx}{\mathbf{v}_p}$, and

$$
||\nabla f(x_p)||^2_{\Sigma_p} = ||\nabla f(x_p)||^2 \frac{\mathbf{v}_p^{2/d}}{12}.
$$

More generally, when $f$ is $\alpha-$differentiable, i.e. that $\forall a \in \mathcal{X}, \exists \nabla^\alpha f(a, \cdot) \in \mathbb{S}_d(0,1)^{\mathbb{R}}$ such that $\forall x \in \mathbb{S}_d(0,1)$, $\lim_{h \to 0} \frac{f(a+hx)-f(a)}{h^\alpha} = \nabla^\alpha f(a, x)$, then it is not too difficult to show that for such hyper-cubic regions, we have

$$
\mathbb{V}_{\mu_p}\left[f(X)\right] = O\left(\mathbf{v}_p^{\frac{2\alpha}{d}} \sup_{\mathbb{S}(0,1)} |\nabla^\alpha f(x_p, u)|^2\right).
$$

On the other hand, by direct computation, the second term is such that $\lim_{\mathbf{v}_p \to 0} \mathbb{E}_{\mu_p}\left[\sigma^2(X)\right] = \sigma^2(x_p)$. Thus, while $\mathbb{V}_{\mu_p}\left[f(X)\right]$ vanishes, $\mathbb{E}_{\mu_p}\left[\sigma^2(X)\right]$ stays bounded away from 0 (unless $\nu$ is deterministic).

## 2.2 Oracle allocation and homogeneous partitioning for piecewise constant mean-approximation.

We now assume that we are allowed to choose the partition $\mathcal{P}$ depending on $n$, thus $\mathcal{P} = \mathcal{P}_n$, amongst all homogeneous partitions of the space, i.e. partitions such that all cells have the same volume, and come from a regular grid of the space. Thus the only free parameter is the number of cells $P_n$ of the partition.

**An exact yet not explicit oracle algorithm.** The minimization of the pseudo-loss (1) does not yield to a closed-form solution in general. However, we can still derive the order of the optimal loss (see [Carpentier and Maillard, 2012, Lemma 2] in the full version of the paper for an example of minimax yet non adaptive oracle algorithm given in closed-form solution):

**Lemma 1** *In the case when $\mathbb{V}_{\mu_p}\big[f(X)\big] = \Omega\big(P_n^{-\alpha'}\big)$ and $\int_{\mathcal{R}_p} \sigma^2(x)\mu_p(dx) = \Omega\big(P_n^{-\beta'}\big)$, then an optimal allocation and partitioning strategy $\mathcal{A}_n^\star$ satisfies that*

$$P_n^\star = \Omega(n^{\frac{1}{\max(1+\alpha'-\beta',1)}}) \quad and \quad T_{p,n}^\star \overset{\text{def}}{=} \frac{\mathbb{V}_{\mu_p}\big[f(X)\big] + \mathbb{E}_{\mu_p}\big[\sigma^2(X)\big]}{L - \mathbb{V}_{\mu_p}\big[f(X)\big]},$$

*as soon as there exists, for such range of $P_n^\star$, a constant $L$ such that*

$$\sum_{p=1}^{P_n^\star} \frac{\mathbb{V}_{\mu_p}\big[f(X)\big] + \mathbb{E}_{\mu_p}\big[\sigma^2(X)\big]}{L - \mathbb{V}_{\mu_p}\big[f(X)\big]} = n\,.$$

*The pseudo-loss of such an algorithm $\mathcal{A}_n^\star$, optimal amongst the allocations strategies that use the partition $\mathcal{P}_n$ in $P_n^\star$ regions, is then given by*

$$\mathcal{L}_n(\mathcal{A}_n^\star) = \Omega\big(n^\gamma\big) \quad where \quad \gamma \overset{\text{def}}{=} \frac{\max(1-\beta', 1-\alpha')}{\max(1+\alpha'-\beta', 1)} - 1\,.$$

The condition involving the constant $L$ is here to ensure that the partition is not degenerate. It is morally satisfied as soon as the variance of $f$ and the noise are bounded and $n$ is large enough.

This Lemma applies to the important class $\mathcal{W}^{1,2}(\mathbb{R})$ of functions that admit a weak derivative that belongs to $L_2(\mathbb{R})$. Indeed these functions are Hölder with coefficient $\alpha = 1/2$, i.e. we have $\mathcal{W}^{1,2}(\mathbb{R}) \subset \mathcal{C}^{0,1/2}(\mathbb{R})$. The standard Brownian motion is an example of function that is $1/2$-Hölder. More generally, for $k = \frac{d}{2} + \alpha$ with $\alpha = 1/2$ when $d$ is odd and $\alpha = 1$ when $d$ is even, we have the inclusion

$$\mathcal{W}^{k,2}(\mathbb{R}^d) \subset \mathcal{C}^{0,\alpha}(\mathbb{R}^d)\,,$$

where $\mathcal{W}^{k,2}(\mathbb{R}^d)$ is the set of functions that admit a $k^{\text{th}}$ weak derivative belonging to $L_2(\mathbb{R}^d)$. Thus the previous Lemma applies to sufficiently smooth functions with smoothness linearly increasing with the dimension $d$ of the input space $\mathcal{X}$.

**Important remark** Note that this Lemma gives us a choice of the partition that is minimax-optimal, and an allocation strategy on that partition that is not only minimax-optimal but also adaptive to the function $f$ itself. Thus it provides a way to decide in a minimax way what is the good number of regions, and then to provide the best oracle way to allocate the budget.

We can deduce the following immediate corollary on the class of $\alpha-$Hölder functions observed in a non-negligible noise of bounded variance (i.e. in the setting $\beta' = 0$ and $\alpha' = \frac{2\alpha}{d}$).

**Corollary 1** *Consider that $f$ is $\alpha-$Hölder and the noise is of bounded variance. Then a minimax-optimal partition satisfies $P_n^\star = \Omega(n^{\frac{d}{d+2\alpha}})$ and an optimal allocation achieves the rate $\mathcal{L}_n(\mathcal{A}_n^\star) = \Omega\big(n^{\frac{-2\alpha}{d+2\alpha}}\big)$. Moreover, the strategy of Lemma 1 is optimal amongst the allocations strategies that use the partition $\mathcal{P}_n$ in $P_n^\star$ regions.*

The rate $\Omega\big(n^{\frac{-2\alpha}{d+2\alpha}}\big)$ is minimax-optimal on the class of $\alpha-$Hölder functions (see Györfi et al. [2002], Ibragimov and Hasminski [1981], Stone [1980]), and it is thus interesting to consider an initial number of regions $P_n^\star$ that is of order $P_n^\star = \Omega(n^{\frac{d}{d+2\alpha}})$. After having built the partition, if the quantities $\big\{\mathbb{V}_{\mu_p}\big[f\big]\big\}_{p\leq P}$ and $\big\{\mathbb{E}_{\mu_p}\big[\sigma^2\big]\big\}_{p\leq P}$ are known to the learner, it is optimal, in the aim of minimizing the pseudo-loss, to allocate to each region the number of samples $T_{p,n}^\star$ provided in Lemma 1. Our objective in this paper is, after having chosen beforehand a minimax-optimal partition, to allocate

the samples properly in the regions, without having any access to those quantities. It is then necessary to balance between exploration, i.e. allocating the samples in order to estimate $\left\{ \mathbb{V}_{\mu_p}[f] \right\}_{p \leq P}$ and $\left\{ \mathbb{E}_{\mu_p}[\sigma^2] \right\}_{p \leq P}$, and exploitation, i.e. use the estimates to target the optimal allocation.

# 3 Online algorithms for allocation and homogeneous partitioning for piecewise constant mean-approximation

In this section, we now turn to the design of algorithms that are fully online, with the goal to be competitive against the kind of oracle algorithms considered in Section 2.2. We now assume that the space $\mathcal{X} = [0,1]^d$ is divided in $P_n$ hyper-cubic regions of same measure (the Lebesgue measure on $[0,1]^d$) $\mathbf{v}_p = \mathbf{v} = \frac{1}{P_n}$. The goal of an algorithm is to minimize the quadratic error of approximation of $f$ by a constant over each cell, in *expectation*, which we write as

$$\max_{1 \leq p \leq P_n} \mathbb{E}\left[ \int_{\mathcal{R}_p} (f(x) - \hat{f}_n(x))^2 \frac{\lambda(dx)}{\lambda(\mathcal{R}_p)} \right] = \max_{1 \leq p \leq P_n} \mathbb{E}\left[ \int_{\mathcal{R}_p} (f(x) - \hat{m}_{p,n})^2 \frac{\lambda(dx)}{\lambda(\mathcal{R}_p)} \right],$$

where $\hat{f}_n$ is the histogram estimate of the function $f$ on the partition $\mathcal{P}$ and $\hat{m}_{p,n}$ is the empirical mean defined on region $\mathcal{R}_p$ with the samples $(X_i, Y_i)$ such that $X_i \in \mathcal{R}_p$. To do so, an algorithm is only allowed to specify at each time step $t$, the next point $X_t$ where to sample, based on all the past samples $\{(X_s, Y_s)\}_{s < t}$. The total budget $n$ is known at the beginning as well as $P_n$ and the regions $\{\mathcal{R}_p\}_{1 \leq p \leq P_n}$.

We want to compare the strategy of an online learning algorithm to the strategy of an oracle that perfectly knows the law $\nu$. We however restrict the power of the oracle by forcing it to only sample uniformly inside a region $\mathcal{R}_p$. Thus the oracle is only allowed to choose at each time step $t$ in which cell $\mathcal{R}_p$ to sample, but is not allowed to decide which point in the cell it can sample. The point $X_t$ has to be sampled uniformly in $\mathcal{R}_p$.

Now, since a learning algorithm has no access to the true distribution $\nu$, we give slightly more power to the learning algorithm by allowing it to resort to a refined partition. We allow it to divide each region $\mathcal{R}_p$ for $p \in \{1, \dots, P_n\}$ into $K$ hyper-cubic sub-regions $\{\mathcal{R}_{p,k}\}_{1 \leq k \leq K}$ of same Lebesgue measure, resulting in a total number $P_n^+ \overset{\text{def}}{=} K P_n$ of hyper-cubic regions of same measure $\mathbf{v}_{p,k} = \frac{1}{K P_n}$. Equivalently, this can be seen as letting the player use a refined partition with $P_n^+$ cells. However, instead of sampling one point in $\mathcal{R}_{p,k}$, the algorithm is only allowed to sample all the $K$ points in region in the chosen $\mathcal{R}_p$ at the same time, one uniformly in each sub-region $\mathcal{R}_{p,k}$, still using of course the same total budget of $n$ points (and not $nK$). Thus the algorithm is free to choose $K$, but once a region $\mathcal{R}_p$ is chosen at time $t$, it can not choose moreover which point to sample inside that region but only sample a set of points in one shot. The reason to do so is that this will allow us to estimate the unknown quantities such as the quadratic variation of $f$ on each region, but we do not want to give the learner too much power. This one shot restriction is also for clarity purpose, as otherwise one has to consider technical details and perform nasty computations that in the end only affects second order terms. The effect of the factor $K$ on the performance bound can be seen in Section 4. For $P_n$ of minimax order, our result shows that $K$ can be chosen to be a (large) constant.

## 3.1 Assumptions

In order to derive performance bounds for a learning algorithm that does not know the noise and the local variance of the function, we now need some assumptions on the data. These are here to ensure that concentration properties apply and that empirical moments are close to true moments with high probability depending on the number of samples. These add to the two other assumptions on the structure of the histograms (uniformed grid partitions) and on the active scheme (that is we can choose a bean but only get a random sample uniformly distributed in that bean).

We assume that $\nu$ is exactly sub-Gaussian, meaning that for all $x \in \mathcal{X}$, the variance of the **noise**$(x)$, written $\sigma^2(x) < \infty$ satisfies that

$$\forall \lambda \in \mathbb{R}^+ \quad \log \mathbb{E} \exp[\lambda \, \mathbf{noise}(x)] \leq \frac{\lambda^2 \sigma^2(x)}{2},$$

and we further assume that it satisfies the following slightly stronger second property (that is for instance exactly verified for a Gaussian variable, looking at the moment generating function):

$$\forall \lambda, \gamma \in \mathbb{R}^+ \quad \log \mathbb{E} \exp\left[\lambda \mathbf{noise}(x) + \gamma \mathbf{noise}(x)^2\right] \leq \frac{\lambda^2 \sigma^2(x)}{2(1 - 2\gamma \sigma^2(x))} - \frac{1}{2} \log\left(1 - 2\gamma \sigma^2(x)\right).$$

The function $f$ is assumed to be $(L, \alpha)$-Hölder, meaning that it satifies
$$\forall x, x' \in \mathcal{X} \;\; f(x) - f(x') \leq L||x - x'||^{\alpha} \,.$$
Similarly, the function $\sigma^2$ is assumed to be $(M, \beta)$-Hölder i.e. it satisfies
$$\forall x, x' \in \mathcal{X} \;\; \sigma^2(x) - \sigma^2(x') \leq M||x - x'||^{\beta} \,.$$

We assume that $\mathcal{Y}$ is a convex and compact subset of $\mathbb{R}$, thus w.l.g. that it is $[0, 1]$, and that it is known that $||\sigma^2||_{\infty}$, which is thus finite, is bounded by the constant 1.

## 3.2 Empirical estimation of the quadratic approximation error on each cell

We define the sampling distribution $\tilde{\mu}_p$ in the region $\mathcal{R}_p$ for each $p \in \{1, \ldots, P_n\}$ as a quasi-uniform sampling scheme using the uniform distribution over the sub-regions. More precisely at time $t \leq n$, if we decide to sample in the region $\mathcal{R}_p$ according to $\tilde{\mu}_p$, we sample uniformly in each sub-region one sample, resulting in a new batch of samples $\{(X_{t,k}, Y_{t,k})\}_{1 \leq k \leq K}$, where $X_{t,k} \sim \mu_{p,k}$. Note that due to this sampling process, the number of points $T_{p,t}$ sampled in sub-region $\mathcal{R}_p$ at time $t$ is always a multiple of $K$ and that moreover for all $k, k' \in \{1, \ldots, K\}$ we have that $T_{p,k,t} = T_{p,k',t} = \frac{T_{p,t}}{K}$. Now this specific sampling is used in order to be able to estimate the variances $\mathbb{V}_{\mu_p} f$ and $\mathbb{E}_{\mu_p} \sigma^2$, so that the best proportions $T_{p,n}^{\star}$ can be computed as accurately as possible. Indeed, as explained in Lemma 1, we have that
$$T_{p,n}^{\star} \stackrel{\text{def}}{=} \frac{\mathbb{V}_{\mu_p}\left[f(X)\right] + \mathbb{E}_{\mu_p}\left[\sigma^2(X)\right]}{L - \mathbb{V}_{\mu_p}\left[f(X)\right]} \,.$$

**Variance estimation** We now introduce two estimators. The first estimator is written $\hat{\mathbf{V}}_{p,t}$ and is built in the following way. First, let us introduce the empirical estimate $\hat{f}_{p,k,t}$ of the mean $f_{p,k} \stackrel{\text{def}}{=} \mathbb{E}_{\mu_{p,k}}\left[f(X)\right]$ of $f$ in sub-region $\mathcal{R}_{p,k}$. Similarly, to avoid some cumbersome notations, we introduce $f_p \stackrel{\text{def}}{=} \mathbb{E}_{\mu_p}\left[f(X)\right]$ and $v_{p,k} \stackrel{\text{def}}{=} \mathbb{V}_{\mu_{p,k}}\left[f(X)\right]$ for the function $f$, and then $\sigma_{p,k}^2 \stackrel{\text{def}}{=} \mathbb{E}_{\mu_{p,k}}\left[\sigma^2(X)\right]$ for the variance of the noise $\sigma^2$. We now define the empirical variance estimator to be
$$\hat{\mathbf{V}}_{p,t} = \frac{1}{K-1} \sum_{k=1}^{K} (\hat{f}_{p,k,t} - \hat{m}_{p,t})^2 \,,$$

that is a biased estimator. Indeed, for a deterministic $T_{p,t}$, it is not difficult to show that we have
$$\mathbb{E}\left[\hat{\mathbf{V}}_{p,t}\right] = \frac{1}{K-1} \sum_{k=1}^{K} \left(\mathbb{E}_{\mu_{p,k}}[f] - \mathbb{E}_{\mu_p}[f]\right)^2 + \frac{1}{T_{p,t}} \sum_{k=1}^{K} \left(\mathbb{V}_{\mu_{p,k}}\left[f\right] + \mathbb{E}_{\mu_{p,k}}\left[\sigma^2\right]\right) \,.$$
The leading term in this decomposition, that is given by the first sum, is closed to $\mathbb{V}_{\mu_p}\left[f\right]$ since, by using the assumption that $f$ is $(L, \alpha)$-Hölder, we have the following inequality
$$\left| \frac{1}{K} \sum_{k=1}^{K} \left(\mathbb{E}_{\mu_{p,k}}[f] - \mathbb{E}_{\mu_p}[f]\right)^2 - \mathbb{V}_{\mu_p}\left[f(X)\right] \right| \leq \frac{2L^2 d^{\alpha}}{(KP_n)^{2\alpha/d}} \,,$$

where we also used that the diameter of a sub-region $\mathcal{R}_{p,k}$ is given by $\mathbf{diam}(\mathcal{R}_{p,k}) = \frac{d^{1/2}}{(KP_n)^{1/d}}$. Then, the second term also contributes to the bias, essentially due to the fact that $\mathbb{V}[\hat{f}_{p,k,t}] = \frac{1}{T_{p,k,t}}(v_{p,k} + \sigma_{p,k}^2)$ and not $\frac{1}{T_{p,t}}(v_k + \sigma_k^2)$ (with $v_p \stackrel{\text{def}}{=} \mathbb{V}_{\mu_p}\left[f(X)\right]$ and $\sigma_p^2 \stackrel{\text{def}}{=} \mathbb{E}_{\mu_p}\left[\sigma^2(X)\right]$).

In order to correct this term, we now introduce the second estimator $\hat{\sigma}_{p,k,t}^2$ that estimates the variance of the outputs in a region $\mathcal{R}_{p,k}$, i.e. $\mathbb{V}_{\mu_{p,k},\nu}\left[Y(X)\right] = \mathbb{V}_{\mu_{p,k}}\left[f(X)\right] + \mathbb{E}_{\mu_{p,k}}\left[\sigma^2\right]$. It is defined as
$$\hat{\sigma}_{p,k,t}^2 \stackrel{\text{def}}{=} \frac{1}{T_{p,k,t} - 1} \sum_{i=1}^{t} \left(Y_i - \frac{1}{T_{p,k,t}} \sum_{j=1}^{t} Y_j \mathbb{I}\{X_j \in \mathcal{R}_{p,k}\}\right)^2 \mathbb{I}\{X_i \in \mathcal{R}_{p,k}\} \,.$$

Now, we combine the two previous estimators to form the following estimator
$$\hat{\mathbf{Q}}_{p,t} = \hat{\mathbf{V}}_{p,t} - \frac{1}{K} \sum_{k=1}^{K} \left(\frac{1}{T_{p,k,t}} - \frac{1}{T_{p,t}}\right) \hat{\sigma}_{p,k,t}^2 \,.$$

The following proposition provides a high-probability bound on the difference between $\hat{\mathbf{Q}}_{p,t}$ and the quantity we want to estimate. We report the detailed proof in [Carpentier and Maillard, 2012].

**Proposition 1** *By the assumption that $f$ is $(L, \alpha)$-Hölder, the bias of the estimator $\hat{\mathbf{Q}}_{p,t}$, and for deterministic $T_{p,t}$, is given by*

$$\mathbb{E}\left[\hat{\mathbf{Q}}_{p,t} - Q_p(T_{p,t})\right] = \frac{1}{K}\sum_{k=1}^{K}\left(\mathbb{E}_{\mu_{p,k}}[f] - \mathbb{E}_{\mu_p}[f]\right)^2 - \mathbb{V}_{\mu_p}\left[f(X)\right] \leq \frac{2L^2 d^\alpha}{(KP_n)^{2\alpha/d}}\,.$$

*Moreover, it satisfies that for all $\delta \in [0,1]$, there exists an event of probability higher than $1 - \delta$ such that on this event, we have*

$$\left|\hat{\mathbf{Q}}_{p,t} - \mathbb{E}\left[\hat{\mathbf{Q}}_{p,t}\right]\right| \leq \sqrt{\frac{8\log(4/\delta)}{(K-1)^2}\sum_{k=1}^{K}\frac{\hat{\sigma}^2_{p,k,t}}{T^2_{p,k,t}}} + o\left(\frac{1}{T_{p,k,t}\sqrt{K}}\sqrt{\frac{1}{K}\sum_{k=1}^{K}\sigma^2_{p,k}}\right)\,.$$

We also state the following Lemma that we are going to use in the analysis, and that takes into account randomness of the stopping times $T_{p,k,t}$.

**Lemma 2** *Let $\{X_{p,k,u}\}_{p \leq P,\, k \leq K,\, u \leq n}$ be samples potentially sampled in region $\mathcal{R}_{p,k}$. We introduce $q_{p,u}$ to be the equivalent of $Q_p(T_{p,t})$ with explicitly fixed value of $T_{p,t} = u$. Let also $\hat{q}_{p,u}$ be the estimate of $\mathbb{E}\left[q_{p,u}\right]$, that is to say the equivalent of $\hat{\mathbf{Q}}_{p,t}$ but computed with the first $u$ samples in each region $\mathcal{R}_{p,k}$ (i.e. $T_{p,t} = u$). Let us define the event*

$$\xi_{n,P,K}(\delta) = \bigcap_{p \leq P}\bigcap_{u \leq n}\left\{\omega : \left|\hat{q}_{p,u}(\omega) - \mathbb{E}\left[q_{p,u}\right]\right| \leq \frac{AK}{u}\sqrt{\frac{\log(4nP/\delta)\hat{V}_{p,t}}{K-1}} + \frac{2L^2 d^\alpha}{(KP_n)^{2\alpha/d}}\right\},$$

*where $\hat{V}_{p,t} = \hat{V}_p(T_{p,t}) = \frac{1}{K-1}\sum_{k=1}^{K}\hat{\sigma}^2_{p,k,t}$ and where $A \leq 4$ is a numerical constant. Then it holds that*

$$\mathbb{P}\left(\xi_{n,P,K}(\delta)\right) \geq 1 - \delta\,.$$

Note that, with the notations of this Lemma, Proposition 1 above is thus about $\hat{q}_{p,u}$.

### 3.3 The Online allocation and homogeneous partitioning algorithm for piecewise constant mean-approximation (OAHPA-pcma)

We are now ready to state the algorithm that we propose for minimizing the quadratic error of approximation of $f$. The algorithm is described in Figure 1. Although it looks similar, this algorithm is quite different from a normal UCB algorithm since $\hat{\mathbf{Q}}_{p,t}$ decreases in expectation with $T_{p,t}$. Indeed, its expectation is close to $\mathbb{V}_{\mu_p}\left[f\right] + \frac{1}{KT_{p,t}}\sum_{k=1}^{K}\left(\mathbb{V}_{\mu_{p,k}}\left[f\right] + \mathbb{E}_{\mu_{p,k}}\left[\sigma^2\right]\right)$.

---

**Algorithm 1** OAHPA-pcma.

---

1: **Input:** $A$, $L$, $\alpha$, Horizon $n$; Partition $\{\mathcal{R}_p\}_{p \leq P}$, with sub-partitions $\{\mathcal{R}_{p,k}\}_{k \leq K}$.
2: **Initialization:** Sample $K$ points in every sub-region $\{\mathcal{R}_{p,k}\}_{p \leq P, k \leq K}$
3: **for** $t = K^2 P + 1$; $t \leq n$; $t = t + K$ **do**
4:   Compute $\forall p, \hat{\mathbf{Q}}_{p,t}$.
5:   Compute $\forall p, B_{p,t} = \hat{\mathbf{Q}}_{p,t} + \frac{AK}{T_{p,t}}\sqrt{\frac{\log(4nP/\delta)\hat{V}_{p,t}}{K-1}} + \frac{2L^2 d^\alpha}{(KP_n)^{2\alpha/d}}$.
6:   Select the region $p_t = \operatorname{argmax}_{1 \leq p \leq P_n} B_{p,t}$ where to sample.
7:   Sample $K$ samples in region $\mathcal{R}_{p_t}$ one per sub-region $\mathcal{R}_{p_t,k}$ according to $\mu_{p_t,k}$.
8: **end for**

---

## 4 Performance of the allocation strategy and discussion

Here is the main result of the paper; see the full version [Carpentier and Maillard, 2012] for the proof. We remind that the objective is to minimize for an algorithm $\mathcal{A}$ the pseudo-loss $\mathcal{L}_n(\mathcal{A})$.

**Theorem 1 (Main result)** *Let $\gamma = \frac{\max_p T^\star_{p,n}}{\min_p T^\star_{p,n}}$ be the distortion factor of the optimal allocation strategy, and let $\epsilon > 0$. Then with the choice of the number of regions $P_n \stackrel{\text{def}}{=} n^{\frac{d}{2\alpha+d}}\epsilon^{2+\frac{d}{2\alpha}}$, and of the number of sub-regions $K \stackrel{\text{def}}{=} C^{\frac{2d}{4\alpha+d}}\epsilon^{-2-\frac{d}{\alpha}}$, where $C \stackrel{\text{def}}{=} \frac{8L^2\alpha}{Ad^{1-\alpha}}$ then the pseudo-loss of the OAHPA-pcma algorithm satisfies, under the assumptions of Section 3.1 and on an event of probability higher than $1 - \delta$,*

$$\mathcal{L}_n(\mathcal{A}) \leq \left(1 + \epsilon\gamma C'\sqrt{\log(1/\delta)}\right)\mathcal{L}_n(\mathcal{A}^\star_n) + o\left(n^{-\frac{2\alpha}{2\alpha+d}}\right),$$

*for some numerical constant $C'$ not depending on $n$, where $\mathcal{A}^\star_n$ is the oracle of Lemma 1.*

**Minimax-optimal partitioning and $\epsilon$-adaptive performance** Theorem 1 provides a high probability bound on the performance of the OAHPA-pcma allocation strategy. It shows that this performance is competitive with that of an optimal (i.e. adaptive to the function $f$, see Lemma 1) allocation $\mathcal{A}^\star$ on a partition with a number of cells $P_n$ chosen to be of minimax order $n^{\frac{d}{2\alpha+d}}$ for the class of $\alpha$-Hölder functions. In particular, since $\mathcal{L}_n(\mathcal{A}_n^\star) = O(n^{\frac{2\alpha}{d+2\alpha}})$ on that class, we recover the same minimax order as what is obtained in the batch learning setting, when using for instance wavelets, or Kernel estimates (see e.g. Stone [1980], Ibragimov and Hasminski [1981]). But moreover, due to the adaptivity of $\mathcal{A}_n^\star$ to the function itself, this procedure is also $\epsilon$-adaptive to the function and not only minimax-optimal on the class, on that partition (see Section 2.2). Naturally, the performance of the method increases, in the same way than for any classical functional estimation method, when the smoothness of the function increases. Similarly, in agreement with the classical curse of dimension, the higher the dimension of the domain, the less efficient the method.

**Limitations** In this work, we assume that the smoothness $\alpha$ of the function is available to the learner, which enables her to calibrate $P_n$ properly. Now it makes sense to combine the OAHPA-pcma procedure with existing methods that enable to estimate this smoothness online (under a slightly stronger assumption than Hölder, such as Hölder functions that attain their exponents, see Giné and Nickl [2010]). It is thus interesting, when no preliminary knowledge on the smoothness of $f$ is available, to spend some of the initial budget in order to estimate $\alpha$.

We have seen that the OAHPA-pcma procedure, although very simple, manages to get minimax optimal results. Now the downside of the simplicity of the OAHPA-pcma strategy is two-fold.
The first limitation is that the factor $(1 + \epsilon\gamma C'\sqrt{\log(1/\delta)}) = (1 + O(\epsilon))$ appearing in the bound before $\mathcal{L}_n(\mathcal{A}^\star)$ is not 1, but higher than 1. Of course it is generally difficult to get a constant 1 in the batch setting (see Arlot [2007]), and similarly this is a difficult task in our online setting too: If $\epsilon$ is chosen to be small, then the error with respect to the optimal allocation is small. However, since $P_n$ is expressed as an increasing function of $\epsilon$, this implies that the minimax bound on the loss for partition $\mathcal{P}$ increases also with $\epsilon$. That said, in the view of the work on active learning multi-armed bandit that we extend, we would still prefer to get the optimal constant 1.
The second limitation is more problematic: since $K$ is chosen irrespective of the region $\mathcal{R}_p$, this causes the presence of the factor $\gamma$. Thus the algorithm will essentially no longer enjoy near-optimal performance guarantees when the optimal allocation strategy is highly not homogeneous.

**Conclusion and future work** In this paper, we considered online regression with histograms in an active setting (we select in which bean to sample), and when we can choose the histogram in a class of homogeneous histograms. Since the (unknown) noise is heteroscedastic and we compete not only with the minimax allocation oracle on $\alpha$-Hölder functions but with the adaptive oracle that uses a minimax optimal histogram and allocates samples adaptively to the target function, this is an extremely challenging (and very practical) setting. Our contribution can be seen as a non trivial extension of the setting of active learning for multi-armed bandits to the case when each arm corresponds to one continuous region of a sampling space, as opposed to a singleton, which can also be seen as a problem of non parametric function approximation. This new setting offers interesting challenges: We provided a simple procedure, based on the computation of upper confidence bounds of the estimation of the local quadratic error of approximation, and provided a performance analysis that shows that OAHPA-pcma is first order $\epsilon$-optimal with respect to the function, for a partition chosen to be minimax-optimal on the class of $\alpha$-Hölder functions. However, this simplicity also has a drawback if one is interested in building exactly first order optimal procedure, and going beyond these limitations is definitely not trivial: A more optimal but much more complex algorithm would indeed need to tune a different factor $K_p$ in each cell in an online way, i.e. define some $K_{p,t}$ that evolves with time, and redefine sub-regions accordingly. Now, the analysis of the OAHPA-pcma already makes use of powerful tools such as empirical-Bernstein bounds for variance estimation (and not only for mean estimation), which make it non trivial; in order to handle possibly evolving sub-regions and deal with the progressive refinement of the regions, we would need even more intricate analysis, due to the fact that we are online and active. This interesting next step is postponed to future work.

**Acknowledgements** This research was partially supported by Nord-Pas-de-Calais Regional Council, French ANR EXPLO-RA (ANR-08-COSI-004), the European Communitys Seventh Framework Programme (FP7/2007-2013) under grant agreement no 270327 (CompLACS) and no 216886 (PASCAL2).

# References

Andràs Antos, Varun Grover, and Csaba Szepesvàri. Active learning in heteroscedastic noise. *Theoretical Computer Science*, 411(29-30):2712–2728, 2010.

Sylvain Arlot. *Rééchantillonnage et Sélection de modèles*. PhD thesis, Université Paris Sud - Paris XI, 2007.

A. Baranes and P.-Y. Oudeyer. R-IAC: Robust Intrinsically Motivated Exploration and Active Learning. *IEEE Transactions on Autonomous Mental Development*, 1(3):155–169, October 2009.

D. Bosq and J.P. Lecoutre. *Théorie de l'estimation fonctionnelle*, volume 21. Economica, 1987.

Alexandra Carpentier and Odalric-Ambrym Maillard. Online allocation and homogeneous partitioning for piecewise constant mean-approximation. *HAL*, 2012. URL http://hal.archives-ouvertes.fr/hal-00742893.

Alexandra Carpentier, Alessandro Lazaric, Mohammad Ghavamzadeh, Rmi Munos, and Peter Auer. Upper-confidence-bound algorithms for active learning in multi-armed bandits. In Jyrki Kivinen, Csaba Szepesvàri, Esko Ukkonen, and Thomas Zeugmann, editors, *Algorithmic Learning Theory*, volume 6925 of *Lecture Notes in Computer Science*, pages 189–203. Springer Berlin / Heidelberg, 2011.

E. Giné and R. Nickl. Confidence bands in density estimation. *The Annals of Statistics*, 38(2): 1122–1170, 2010.

L. Györfi, M. Kohler, A. Krzyźak, and Walk H. *A distribution-free theory of nonparametric regression*. Springer Verlag, 2002.

I. Ibragimov and R. Hasminski. Statistical estimation: Asymptotic theory. 1981.

M. Rosenblatt. *Stochastic curve estimation*, volume 3. Inst of Mathematical Statistic, 1991.

J. Schmidhuber. Formal theory of creativity, fun, and intrinsic motivation (19902010). *Autonomous Mental Development, IEEE Transactions on*, 2(3):230–247, 2010.

C.J. Stone. Optimal rates of convergence for nonparametric estimators. *The annals of Statistics*, pages 1348–1360, 1980.

J.W. Tukey. Non-parametric estimation ii. statistically equivalent blocks and tolerance regions–the continuous case. *The Annals of Mathematical Statistics*, 18(4):529–539, 1947.

